# Integrated Segmentation and Recognition of Hand-Printed Numerals

**James D. Keeler***
MCC
3500 W. Balcones Ctr. Dr.
Austin, TX 78759

**David E. Rumelhart**
Psychology Department
Stanford University
Stanford, CA 94305

**Wee-Kheng Leow**
MCC and
University of Texas
Austin, TX 78759

## Abstract

Neural network algorithms have proven useful for recognition of individual, segmented characters. However, their recognition accuracy has been limited by the accuracy of the underlying segmentation algorithm. Conventional, rule-based segmentation algorithms encounter difficulty if the characters are touching, broken, or noisy. The problem in these situations is that often one cannot properly segment a character until it is recognized yet one cannot properly recognize a character until it is segmented. We present here a neural network algorithm that simultaneously segments and recognizes in an integrated system. This algorithm has several novel features: it uses a supervised learning algorithm (backpropagation), but is able to take position-independent information as targets and self-organize the activities of the units in a competitive fashion to infer the positional information. We demonstrate this ability with overlapping hand-printed numerals.

## 1  INTRODUCTION

A major problem with standard backpropagation algorithms for pattern recognition is that they seem to require carefully segmented and localized input patterns for training. This is a problem for two reasons: first, it is often a labor intensive task to provide this information and second, the decision as to how to segment often depends on prior recognition. However, we describe below a neural network design and corresponding backpropagation learning algorithm that learns to simultane-

ously segment and identify a pattern. [1]

There are two important aspects to many pattern recognition problems that we have built directly into our network and learning algorithm. The first is that the exact location of the pattern, in space or time, is irrelevant to the classification of the pattern; it should be recognized as a member of the same class wherever or whenever it occurs. This suggests that we build translation independence directly into our network. The second aspect is that feedback about *whether* or not a pattern of a particular class is present is all that should be required for training; information about the exact location and relationship to other patterns should not be required. The target information, thus, does not include information about *where* the patterns occur, only about *whether* a particular pattern occurs.

We have incorporated two design principles into our network to deal with these problems. The first is to build translation independence into the network by using linked local receptive fields. The second is to build a fixed "forward model" (c.f. Jordan and Rumelhart, 1990) which translates a location-specific recognition process into a location-independent output value. This output gives rise to a non-specific error signal which is propagated back through this fixed network to train the underlying location-specific network.

## 2   NETWORK ARCHITECTURE AND ALGORITHM

The basic organization of the network is illustrated in Figure 1. In the case of character recognition, the input consists of a set of pixels over which the stimulus patterns are displayed. We designate the stimulus pattern by the vector $\vec{X}$. In general, we assume that any character can be presented in any position and that characters may overlap. The input image then projects to a set of hidden units which learn to abstract features from the input field. These feature abstraction units are organized into sheets, one for each feature type. Each unit within a sheet is constrained to have the same weights as every other unit in the sheet (to enforce translational invariance). This is the same method used by Rumelhart, Hinton and Williams (1986) in solving the so-called T/C problem and the one used by LeCun et al. (1990) in their work on ZIP-code recognition.

We let the activation value of hidden unit of type $i$ at location $j$ be a logistic sigmoidal function of its net input and designate it $h_{ij}$. We interpret $h_{ij}$ as the probability that feature $f_i$ is present in the input at position $j$. The hidden units then project onto a set of sheets of position-specific character recognition units, one sheet for each character type. These units have exponential activation functions and each unit in the sheet receives inputs from a local receptive field block of feature detection units as shown in Figure 1. As with the hidden units, the weights in each exponential unit sheet are linked, enforcing translational invariance. We designate as $\chi_{ij}$ the activation of the unit for detecting character $i$ at location $j$, and define

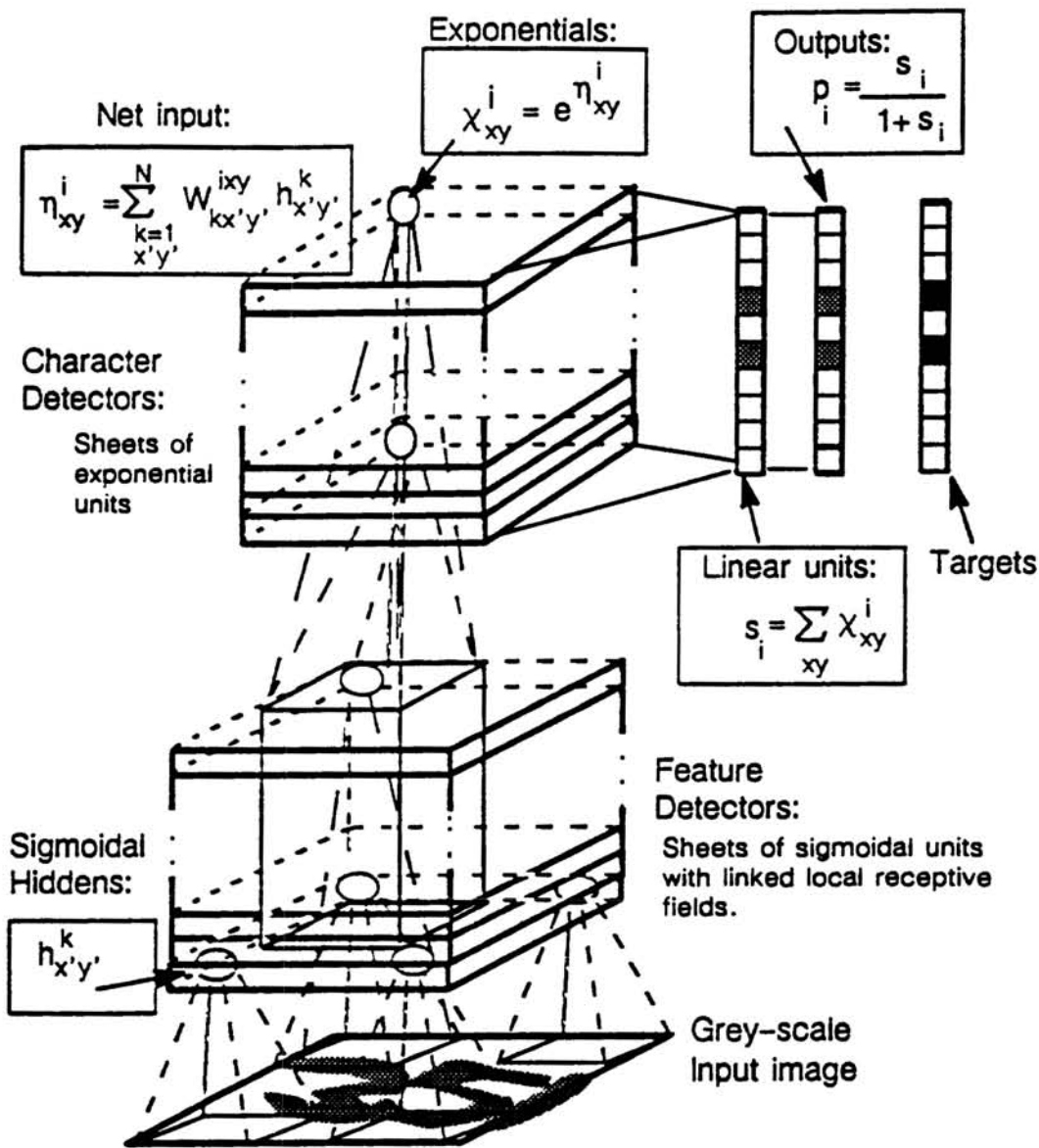

Net input:

$$\eta^{i}_{xy} = \sum_{\substack{k=1 \\ x'y'}}^{N} W^{ixy}_{kx'y'} h^{k}_{x'y'}$$

Exponentials:

$$\chi^{i}_{xy} = e^{\eta^{i}_{xy}}$$

Outputs:

$$p_{i} = \frac{s_{i}}{1+s_{i}}$$

Character
Detectors:

Sheets of
exponential
units

Linear units:

$$s_{i} = \sum_{xy} \chi^{i}_{xy}$$

Targets

Feature
Detectors:

Sheets of sigmoidal units
with linked local receptive
fields.

Sigmoidal
Hiddens:

$$h^{k}_{x'y'}$$

Grey–scale
Input image

Figure 1: The Integrated Segmentation and Recognition (ISR) network. The input image may contain several characters and is presented to the network in a two-dimensional array of grey-scale values. Units in the first block $h^{k}_{x'y'}$ have linked-local receptive fields to the input image, and detect features of type $k$. The exponential units in the next block receive inputs from a local receptive field of hidden sigmoidal units. The weights $W^{ixy}_{kx'y'}$ connect the hidden unit $h^{k}_{x'y'}$ to the exponential unit $\chi^{i}_{xy}$. The architecture enforces translational invariance across the sheets of units by linking the weights and shifting the receptive fields in each dimension. Finally, the activity in each individual sheet of exponential units is summed by the linear units $s_{i}$ and converted to a probability $p_{i}$. The two-dimensional input image can be thought of as a one-dimensional vector $\vec{X}$ as discussed in the text. For notational convenience we used one-dimensional indices (j) in the text rather than two-dimensional (xy) as shown in the figure. All of the mathematics goes through if one replaces $j \leftrightarrow xy$.

$\chi_{ij} = e^{\eta_{ij}}$, where the net input to the unit is

$$\eta_{ij} = \sum_k w_{ik} h_{kj} + \beta_i \qquad (1)$$

and $w_{ik}$ is the weight from hidden unit $h_{kj}$ to the detector $\chi_{ij}$. As we argue in Keeler Rumelhart and Leow (1991), $\eta_{ij}$ can usefully be interpreted as the logarithm of the likelihood ratio favoring the hypothesis that a character of type $i$ is at location $j$ of the input field. Since $\chi_{ij}$ is the exponential of $\eta_{ij}$, the $\chi$ units are to be interpreted as representing the likelihood ratios themselves. Thus, we can interpret the output of the $\chi$ units directly as the *evidence* favoring the assumption that there is a character of a particular type at a particular location. If we were willing and able to carefully segment the input and tell the network the exact location of each character, we could use a standard training technique to train the network to recognize characters at any location with any degree of overlap. However, we are interested in a training algorithm in which we don't have to provide the network with such specific training information. We are interested in simply telling the network which characters are present in the input – not where each character is. This approach saves tremendous time and effort in data preparation and labeling. To implement this idea, we have built an additional network which takes the output of the $\chi$ units and computes, through a fixed output network, the probability that a given character is present anywhere in the input field. We do this by adding two additional layers of units. The first layer of units, the $S$ units, simply sum the activity of each sheet of the $\chi$ units. The activity of unit $S_i$ can, under certain assumptions, be interpreted as the likelihood ratio that a character of type $i$ occurred *anywhere* in the input field. Finally in the output layer, we convert the likelihood ratio into a probability by the formula

$$p_i = \frac{S_i}{1 + S_i}. \qquad (2)$$

Thus, $p_i$ is interpreted as representing directly the probability that character $i$ occurred in the input field.

## 2.1 The learning Rule

On having set up our network, it is straight-forward to compute the derivative of the error function with respect to $\eta_{ij}$. We get a particularly simple learning rule if we let the objective function be the cross-entropy function,

$$l = \sum_i t_i \ln p_i + (1 - t_i) \ln(1 - p_i) \qquad (3)$$

where $t_i$ equals 1 if character $i$ is presented and zero otherwise. In this case, we get the following rule:

$$\frac{\partial l}{\partial \eta_{ij}} = (t_i - p_i) \frac{\chi_{ij}}{\sum_k \chi_{ik}}. \qquad (4)$$

It should be noted that this is a kind of *competitive* rule in which the learning is proportional to the relative strength of the activation at the unit at a particular location in the $\chi$ layer to the strength of activation in the entire layer. This is valid if we assume that either the character appears exactly once or not at all. This ratio is the conditional probability that the target was at position $j$ under the assumption that the target was, in fact, presented. It is also possible to derive a learning rule for the case where more than one of the same character is present[3].

# 3   EXPERIMENTAL RESULTS

To investigate the ability of this network to simultaneously segment and recognize characters in an integrated system, we trained the network outlined in section 2 on a database of hand-printed numerals taken from financial documents. We used a training and test set of about 9,000 and 1,800 characters respectively. We placed pairs of these grey-scaled characters on the input plane at positions determined by a distance parameter which tells how far apart to place the centers of the characters. We used a distance parameter of 1.2 which indicates that the centers were about 1.2 characters apart with an added random displacement in the x and y dimensions by $\pm.25$ and $\pm 0.15$ of the leftmost character size respectively. With these parameters, the characters touch or overlap about 15% of the time. The network had 10 output units and the target was to turn on the units of the two characters in the input window, regardless of what order or position they occurred in. Thus the pair (3,5) has the same target as (5,3): $target = (0001010000)$.

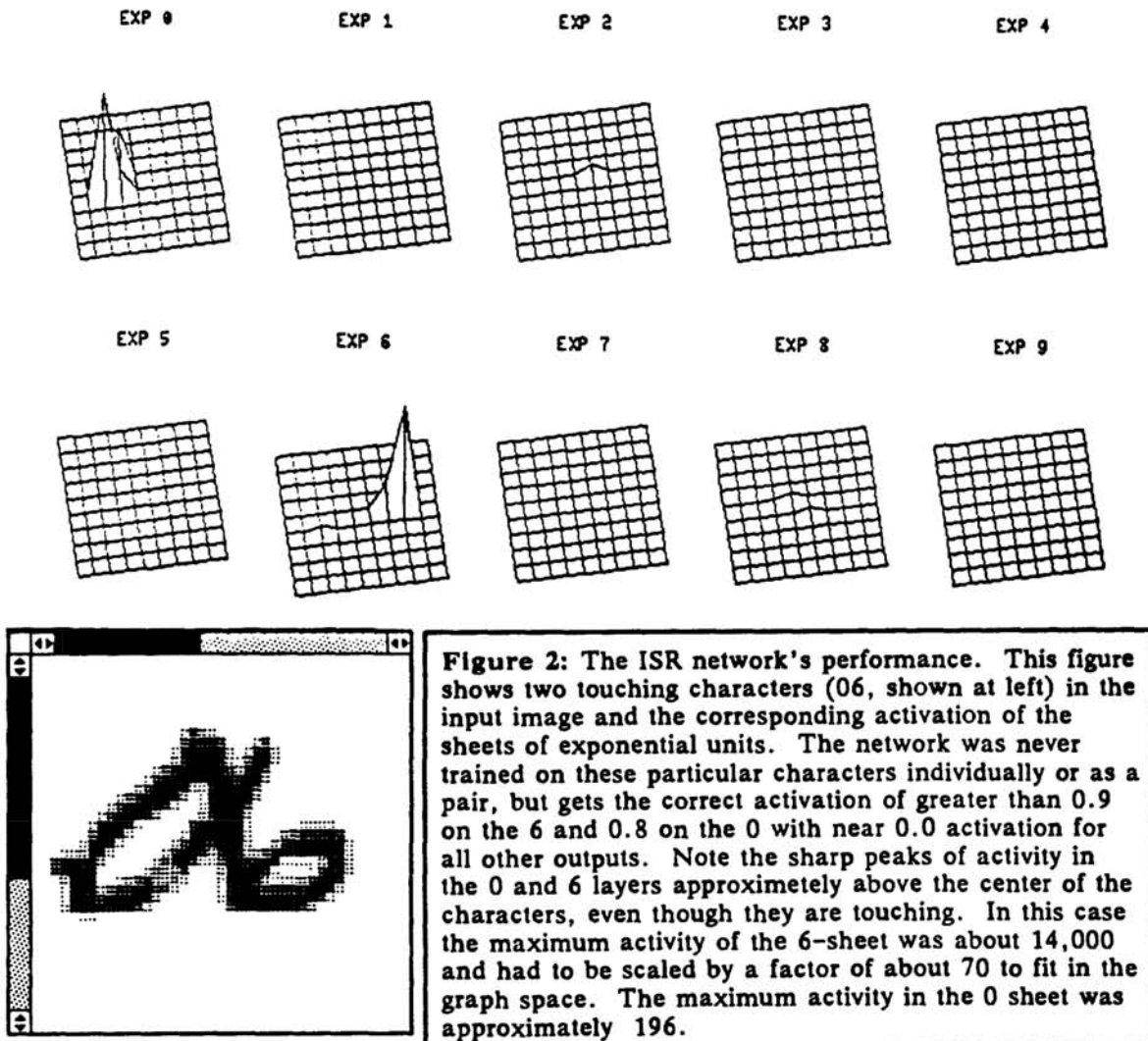

Figure 2: The ISR network's performance. This figure shows two touching characters (06, shown at left) in the input image and the corresponding activation of the sheets of exponential units. The network was never trained on these particular characters individually or as a pair, but gets the correct activation of greater than 0.9 on the 6 and 0.8 on the 0 with near 0.0 activation for all other outputs. Note the sharp peaks of activity in the 0 and 6 layers approximetely above the center of the characters, even though they are touching. In this case the maximum activity of the 6-sheet was about 14,000 and had to be scaled by a factor of about 70 to fit in the graph space. The maximum activity in the 0 sheet was approximately 196.

After training on several hundred thousand of the randomly sampled pairs of numbers from the 9,000, the network generalized correctly on about 81% of the pairs. This pair accuracy corresponds to a single-character recognition accuracy of about 90%. The network recognizes isolated single characters at an accuracy of about 95%. Note that this is an artificially generated data set, and by changing the distance parameter we can make the problem as simple or as difficult as we desire, up to the point where the characters overlap so much that a human cannot recognize them. Most conventional segmentation algorithms do not deal with touching characters, and so would presumably miss the vast majority of these characters. To see how overlap affects performance, we tested generalization in the same network on 100 pairs with the distance parameter lowered to 1.0 and 0.95. With a distance parameter of 1.0, the characters touch or overlap about 50% of the time. Of those, the network correctly identified 80%. Of the 20% that were missed, about 1/2 were unrecognizable by a human. With a distance parameter of 0.95, causing about 66% of the characters to touch, about 74% are correctly identified. As one expects, performance drops for smaller distance parameters.

The qualitative behavior of this system is quite interesting. As described in section 2, the learning rule for the exponential units contains a term that is competitive in nature. This term favors "winner-take-all" behavior for the units in that sheet in the sense that nearly equal activations are unstable under the learning rule: if one presents the same pattern again and again, the learning rule will cause one activation to grow or shrink away from the other at an exponetial rate. This causes self-organization to occur in the exponential sheets, and we would expect the exponential units to organize into highly-localized activations or "spikes" of activity on the appropriate exponential layers directly above the input characters. This is exactly the behavior that is observed in the trained network, as exemplified in Figure 2. In this figure we see two overlapping characters in the input image (06). The network generalized properly with output activity of about 0.8 for zero 0.99 for 6 and about 0.0 for everything else. Note that in the exponential layer, there are very sharp spikes of activity directly above the 0 and the 6 in the appropriate layers. Indeed, it has been our experience that even with quite noisy input images, the representation in the exponential layer is very localized, and we could presumably recover the positional information by examining the activity of the exponential units. We can thus think of these spikes in the exponential layer as "smart histograms": the exponential units in each sheet learn to look for specific combinations of features in the input layer and reject other combinations of inputs. This allows them to respond correctly even if there is a significant amount of noise in the input image, or if the characters happen to be touching or broken.

## 4   DISCUSSION

The system presented here demonstrates that neural networks can, in fact, be used for segmentation as well as recognition. We have by no means demonstrated that this method is better than conventional segmentation/recognition systems in overall performance. However, most conventional systems cannot deal with touching, broken, or noisy characters very well *at all*, whereas the present system handles all of these cases and recognition in a single, integrated fashion. This approach not only offers an integrated solution to the problems at hand, it also has the properties

of being translation invariant, trainable with minimal information, and could be implemented in hardware for extremely fast feed-forward performance.

Note that the architecture discussed here is similar in some respects to the neocognitron model of Fukushima (1980). However, the system is different in several important aspects. First of all, the features here are learned through backpropagation rather than hand-coded as in the neocognitron. Second, the neural network self-organizes positional information via localized activation in the exponential layers. Third, the network is all feed-forward in its run-time dynamics.

Finally, it is worth pointing out that there are other aspects of the problem that we have not dealt with: Our network was trained on approximately the same size characters – to within 40% in height and no normalization in the x-dimension. We have not dealt here with the aspects of normalization, attentional focusing, or recovery of positional information, all of which would be needed in a functioning system.

## Acknowledgements

We thank Peter Robinson from NCR Waterloo for providing the training data and Eric Hartman, Carsten Peterson, Richard Durbin, and Charles Rosenburg for useful discussions.

## Footnotes

*Reprint requests: Jim Keeler, keeler@mcc.com or coila@mcc.com

[1]The algorithm and network design presented here was first proposed by Rumelhart in a presentation entitled "Learning and Generalization in Multilayer networks" given at the NATO Advanced Research Workshop on Neuro Computing, Algorithms, Architectures and Applications held in Les Arcs, France in February, 1989. The algorithm can be viewed as a generalization and refinement of the TDNN of Lang, Hinton, & Waibel, 1990.

## References

[1] K. Fukushima. (1980) Neocognitron: A self-organizing neural network model for a mechanism of pattern recognition unaffected by shift in position. *Biological Cybern.* **36**, 193-202.

[2] M. I. Jordan and D. E. Rumelhart (1990) Forward models: Supervised Learning with a Distal Teacher. *MIT Center for Cognitive Science, Occasional paper #* 40.

[3] J. Keeler, D. E. Rumelhart and W. K. Leow. (1991) Integrated Segmentation and Recognition of Hand-Printed Numerals. *MCC Technical Report ACT-NN-10.91*

[4] K. Lang, A. Waibel and G. Hinton. (1990) A Time Delay Neural Network Architecture for Isolated Word Recognition. *Neural Networks,* **3** 23-44.

[5] Y. Le Cun, B. Boser, J.S. Denker, S. Solla, R. Howard, and L. Jackel. (1990) Back-Propagation applied to Handwritten Zipcode Recognition. *Neural Computation* **1**(4):541-551.

[6] D.E. Rumelhart, G.E.Hinton and R.J.Williams (1986), "Learning Internal Representations by Error Propagation," in D.E.Rumelhart, J.L.McClelland and the PDP Research Group, *Parallel Distributed Processing: Explorations in the Microstructure of Cognition. Volume 1: Foundations,* Cambridge, MA: MIT Press/Bradford.
